# Occam's Razor

**Carl Edward Rasmussen**
Department of Mathematical Modelling
Technical University of Denmark
Building 321, DK-2800 Kongens Lyngby, Denmark
carl@imm.dtu.dk  http://bayes.imm.dtu.dk

**Zoubin Ghahramani**
Gatsby Computational Neuroscience Unit
University College London
17 Queen Square, London WC1N 3AR, England
zoubin@gatsby.ucl.ac.uk  http://www.gatsby.ucl.ac.uk

## Abstract

The Bayesian paradigm apparently only sometimes gives rise to Occam's Razor; at other times very large models perform well. We give simple examples of both kinds of behaviour. The two views are reconciled when measuring complexity of functions, rather than of the machinery used to implement them. We analyze the complexity of functions for some linear in the parameter models that are equivalent to Gaussian Processes, and always find Occam's Razor at work.

## 1 Introduction

Occam's Razor is a well known principle of "parsimony of explanations" which is influential in scientific thinking in general and in problems of statistical inference in particular. In this paper we review its consequences for Bayesian statistical models, where its behaviour can be easily demonstrated and quantified. One might think that one has to build a prior over models which explicitly favours simpler models. But as we will see, Occam's Razor is in fact embodied in the application of Bayesian theory. This idea is known as an "automatic Occam's Razor" [Smith & Spiegelhalter, 1980; MacKay, 1992; Jefferys & Berger, 1992].

We focus on complex models with large numbers of parameters which are often referred to as *non-parametric*. We will use the term to refer to models in which we do not necessarily know the roles played by individual parameters, and inference is not primarily targeted at the parameters themselves, but rather at the predictions made by the models. These types of models are typical for applications in machine learning.

From a non-Bayesian perspective, arguments are put forward for adjusting model complexity in the light of limited training data, to avoid over-fitting. Model complexity is often regulated by adjusting the number of free parameters in the model and sometimes complexity is further constrained by the use of regularizers (such as weight decay). If the model complexity is either too low or too high performance on an independent test set will suffer, giving rise to a characteristic Occam's Hill. Typically an estimator of the generalization error or an independent validation set is used to control the model complexity.

From the Bayesian perspective, authors seem to take two conflicting stands on the question of model complexity. One view is to infer the probability of the model for each of several different model sizes and use these probabilities when making predictions. An alternate view suggests that we simply choose a "large enough" model and sidestep the problem of model size selection. Note that both views assume that parameters are averaged over. Example: Should we use Occam's Razor to determine the optimal number of hidden units in a neural network or should we simply use as many hidden units as possible computationally? We now describe these two views in more detail.

## 1.1 View 1: Model size selection

One of the central quantities in Bayesian learning is the *evidence*, the probability of the data given the model $P(Y|\mathcal{M}_i)$ computed as the integral over the parameters $\mathbf{w}$ of the likelihood times the prior. The evidence is related to the probability of the model, $P(\mathcal{M}_i|Y)$ through Bayes rule:

$$P(Y|\mathcal{M}_i) = \int P(Y|\mathbf{w}, \mathcal{M}_i) P(\mathbf{w}|\mathcal{M}_i) \, d\mathbf{w}, \qquad P(\mathcal{M}_i|Y) = \frac{P(Y|\mathcal{M}_i)P(\mathcal{M}_i)}{P(Y)},$$

where it is not uncommon that the prior on models $P(\mathcal{M}_i)$ is flat, such that $P(\mathcal{M}_i|Y)$ is proportional to the evidence. Figure 1 explains why the evidence discourages overcomplex models, and can be used to select[1] the most probable model.

It is also possible to understand how the evidence discourages overcomplex models and therefore embodies Occam's Razor by using the following interpretation. The evidence is the probability that if you *randomly selected* parameter values from your model class, you would generate data set $Y$. Models that are too simple will be very unlikely to generate that particular data set, whereas models that are too complex can generate many possible data sets, so again, they are unlikely to generate that particular data set at random.

## 1.2 View 2: Large models

In non-parametric Bayesian models there is no *statistical* reason to constrain models, as long as our prior reflects our beliefs. In fact, since constraining the model order (i.e. number of parameters) to some small number would not usually fit in with our prior beliefs about the true data generating process, it makes sense to use large models (no matter how much data you have) and pursue the infinite limit if you can[2]. For example, we ought not to limit the number of basis functions in function approximation a priori since we don't really believe that the data was actually generated from a small number of fixed basis functions. Therefore, we should consider models with as many parameters as we can handle computationally.

Neal [1996] showed how multilayer perceptrons with large numbers of hidden units achieved good performance on small data sets. He used sophisticated MCMC techniques to implement averaging over parameters. Following this line of thought there is no model complexity selection task: We don't need to evaluate evidence (which is often difficult) and we don't need or want to use Occam's Razor to limit the number of parameters in our model.

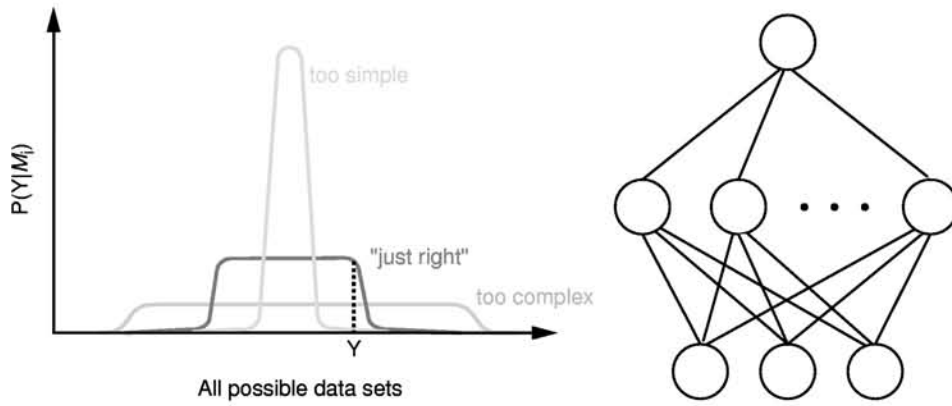

Figure 1: Left panel: the evidence as a function of an abstract one dimensional representation of "all possible" datasets. Because the evidence must "normalize", very complex models which can account for many datasets only achieve modest evidence; simple models can reach high evidences, but only for a limited set of data. When a dataset $Y$ is observed, the evidence can be used to select between model complexities. Such selection cannot be done using just the likelihood, $P(Y|\mathbf{w}, \mathcal{M}_i)$. Right panel: neural networks with different numbers of hidden unit form a family of models, posing the model selection problem.

## 2 Linear in the parameters models – Example: the Fourier model

For simplicity, consider function approximation using the class of models that are linear in the parameters; this class includes many well known models such as polynomials, splines, kernel methods, etc:

$$y(x) = \sum w_i \phi_i(x) \Leftrightarrow \mathbf{y} = \mathbf{w}^\top \Phi,$$

where $y$ is the scalar output, $\mathbf{w}$ are the unknown weights (parameters) of the model, $\phi_i(x)$ are fixed basis functions, $\Phi_{in} = \phi_i(\mathbf{x}^{(n)})$ and $\mathbf{x}^{(n)}$ is the (scalar or vector) input for example number $n$. For example, a Fourier model for scalar inputs has the form:

$$y(x) = a_0 + \sum_{d=1}^{D} a_d \sin(dx) + b_d \cos(dx),$$

where $\mathbf{w} = \{a_0, a_1, b_1, \ldots, a_D, b_D\}$. Assuming an independent Gaussian prior on the weights:

$$p(\mathbf{w}|S, \mathbf{c}) \propto \exp\left(-\frac{S}{2}\left[c_0 a_0^2 + \sum_{d=1}^{D} c_d(a_d^2 + b_d^2)\right]\right),$$

where $S$ is an overall scale and $c_d$ are precisions (inverse variances) for weights of order (frequency) $d$. It is easy to show that Gaussian priors over weights imply Gaussian Process priors over functions[3]. The covariance function for the corresponding Gaussian Process prior is:

$$K(x, x') = \left[\sum_{d=0}^{D} \cos\left(d(x - x')\right)/c_d\right]/S.$$

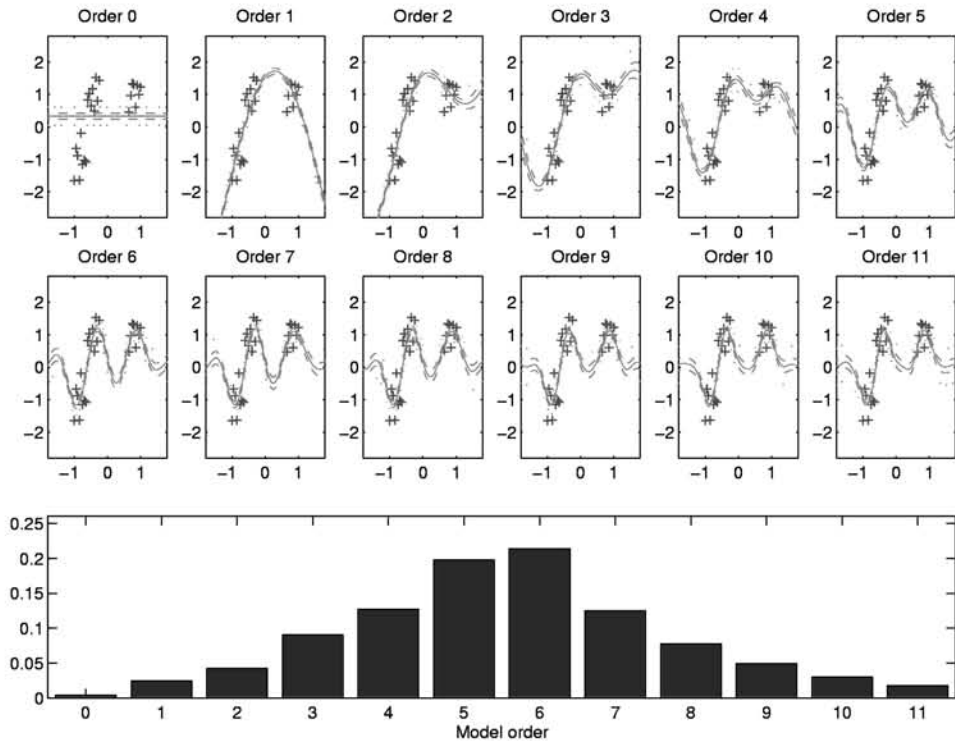

Figure 2: Top: 12 different model orders for the "unscaled" model: $c_d \propto 1$. The mean predictions are shown with a full line, the dashed and dotted lines limit the 50% and 95% central mass of the predictive distribution (which is student-$t$). Bottom: posterior probability of the models, normalised over the 12 models. The probabilities of the models exhibit an Occam's Hill, discouraging models that are either "too small" or "too big".

## 2.1 Inference in the Fourier model

Given data $\mathcal{D} = \{\mathbf{x}^{(n)}, y^{(n)} | n = 1, \ldots, N\}$ with independent Gaussian noise with precision $\tau$, the likelihood is:

$$p(\mathbf{y}|\mathbf{x}, \mathbf{w}, \tau) \propto \prod_{n=1}^{N} \exp\left(-\frac{\tau}{2}[y^{(n)} - \mathbf{w}^\top \Phi_n]^2\right).$$

For analytical convenience, let the scale of the prior be proportional to the noise precision, $S = C\tau$ and put vague[4] Gamma priors on $\tau$ and $C$:

$$p(\tau) \propto \tau^{\alpha_1 - 1}\exp(-\beta_1\tau), \qquad p(C) \propto C^{\alpha_2 - 1}\exp(-\beta_2 C),$$

then we can integrate over weights and noise to get the evidence as a function of prior hyperparameters, $C$ (the overall scale) and $\mathbf{c}$ (the relative scales):

$$E(C, \mathbf{c}) = \iint p(\mathbf{y}|\mathbf{x}, \mathbf{w}, \tau)p(\mathbf{w}|C, \tau, \mathbf{c})p(\tau)p(C)d\tau d\mathbf{w} = \frac{\beta_1^{\alpha_1}\beta_2^{\alpha_2}\Gamma(\alpha_1 + N/2)}{(2\pi)^{N/2}\Gamma(\alpha_1)\Gamma(\alpha_2)}$$

$$\times |\mathbf{A}|^{1/2}\left[\beta_1 + \frac{1}{2}\mathbf{y}^\top(I - \Phi\mathbf{A}^{-1}\Phi^\top)\mathbf{y}\right]^{-\alpha_1 - N/2}C^{D + \alpha_2 - 1/2}\exp(-\beta_2 C)c_0^{1/2}\prod_{d=1}^{D}c_d,$$

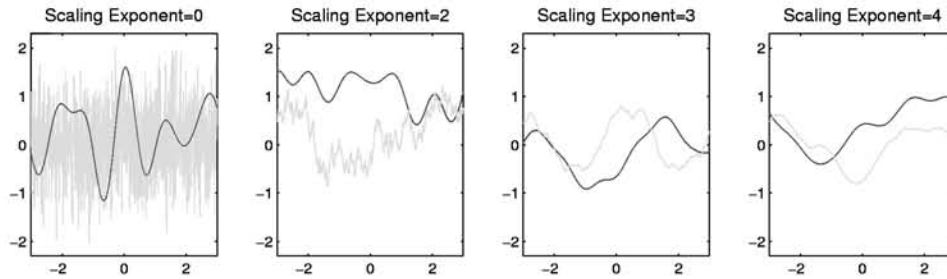

Figure 3: Functions drawn at random from the Fourier model with order $D = 6$ (dark) and $D = 500$ (light) for four different scalings; limiting behaviour from left to right: discontinuous, Brownian, borderline smooth, smooth.

where $\mathbf{A} = \Phi^\top \Phi + C \, \mathrm{diag}(\tilde{\mathbf{c}})$, and the tilde indicates duplication of all components except for the first. We can optimize[5] the overall scale $C$ of the weights (using eg. Newton's method). How do we choose the relative scales, $\mathbf{c}$? The answer to this question turns out to be intimately related to the two different views of Bayesian inference.

## 2.2 Example

To illustrate the behaviour of this model we use data generated from a step function that changes from $-1$ to $1$ corrupted by independent additive Gaussian noise with variance $0.25$. Note that the true function cannot be implemented exactly with a model of finite order, as would typically be the case in realistic modelling situations (the true function is not "realizable" or the model is said to be "incomplete"). The input points are arranged in two lumps of 16 and 8 points, the step occurring in the middle of the larger, see figure 2.

If we choose the scaling precisions to be independent of the frequency of the contributions, $c_d \propto 1$ (while normalizing the sum of the inverse precisions) we achieve predictions as depicted in figure 2. We clearly see an Occam's Razor behaviour. A model order of around $D = 6$ is preferred. One might say that the limited data does not support models more complex than this. One way of understanding this is to note that as the model order grows, the prior parameter volume grows, but the relative posterior volume decreases, because parameters must be accurately specified in the complex model to ensure good agreement with the data. The ratio of prior to posterior volumes is the Occam Factor, which may be interpreted as a penalty to pay for fitting parameters.

In the present model, it is easy to draw functions at random from the prior by simply drawing values for the coefficients from their prior distributions. The left panel of figure 3 shows samples from the prior for the previous example for $D = 6$ and $D = 500$. With increasing order the functions get more and more dominated by high frequency components. In most modelling applications however, we have some prior expectations about smoothness. By scaling the precision factors $c_d$ we can achieve that the prior over functions converges to functions with particular characteristics as $D$ grows towards infinity. Here we will focus on scalings of the form $c_d = d^\gamma$ for different values of $\gamma$, the *scaling exponent*. As an example, if we choose the scaling $c_d = d^3$ we do not get an Occam's Razor in terms of the order of the model, figure 4. Note that the predictions and their errorbars become almost independent of the model order as long as the order is large enough. Note also that the errorbars for these large models seem more reasonable than for $D = 6$ in figure 2 (where a spurious "dip" between the two lumps of data is predicted with high confidence). With this choice of scaling, it seems that the "large models" view is appropriate.

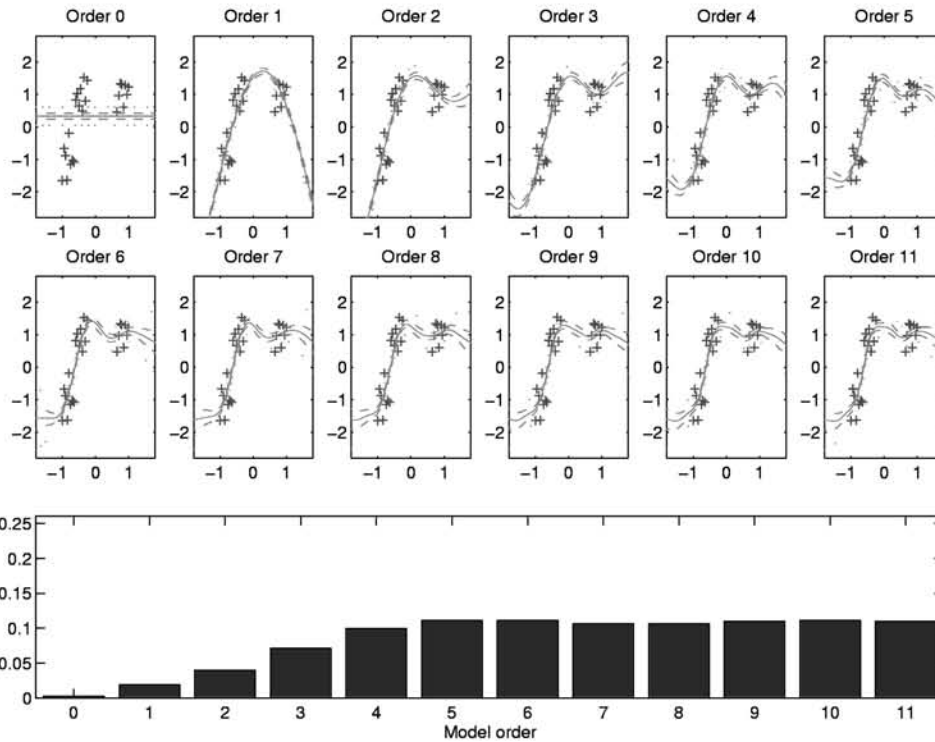

Figure 4: The same as figure 2, except that the scaling $c_d = d^3$ was used here, leading to a prior which converges to smooth functions as $D \to \infty$. There is no Occam's Razor; instead we see that as long as the model is complex enough, the evidence is flat. We also notice that the predictive density of the model is unchanged as long as $D$ is sufficiently large.

## 3  Discussion

In the previous examples we saw that, depending on the scaling properties of the prior over parameters, both the Occam's Razor view and the large models view can seem appropriate. However, the example was unsatisfactory because it is not obvious how to choose the scaling exponent $\gamma$. We can gain more insight into the meaning of $\gamma$ by analysing properties of functions drawn from the prior in the limit of large $D$. It is useful to consider the expected squared difference of outputs corresponding to nearby inputs, separated by $\Delta$:

$$G(\Delta) = E[(f(x) - f(x + \Delta))^2],$$

in the limit as $\Delta \to 0$. In the table in figure 5 we have computed these limits for various values of $\gamma$, together with the characteristics of these functions. For example, a property of smooth functions is that $G(\Delta) \propto \Delta^2$. Using this kind of information may help to choose good values for $\gamma$ in practical applications. Indeed, we can attempt to infer the "characteristics of the function" $\gamma$ from the data. In figure 5 we show how the evidence depends on $\gamma$ and the overall scale $C$ for a model of large order ($D = 200$). It is seen that the evidence has a maximum around $\gamma = 3$. In fact we are seeing Occam's Razor again! This time it is not in terms of the dimension if the model, but rather in terms of the complexity of the functions under the priors implied by different values of $\gamma$. Large values of $\gamma$ correspond to priors with most probability mass on simple functions, whereas small values of $\gamma$ correspond to priors that allow more complex functions. Note, that the "optimal" setting $\gamma = 3$ was exactly the model used in figure 4.

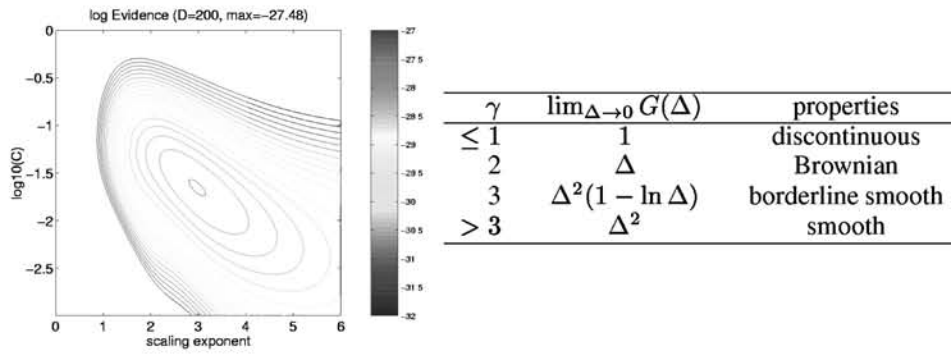

Figure 5: Left panel: the evidence as a function of the scaling exponent, $\gamma$ and overall scale $C$, has a maximum at $\gamma = 3$. The table shows the characteristics of functions for different values of $\gamma$. Examples of these functions are shown in figure 3.

## 4   Conclusion

We have reviewed the automatic Occam's Razor for Bayesian models and seen how, while not necessarily penalising the *number of parameters*, this process is active in terms of the *complexity of functions*. Although we have only presented simplistic examples, the explanations of the behaviours rely on very basic principles that are generally applicable. Which of the two differing Bayesian views is most attractive depends on the circumstances: sometimes the large model limit may be computationally demanding; also, it may be difficult to analyse the scaling properties of priors for some models. On the other hand, in typical applications of non-parametric models, the "large model" view may be the most convenient way of expressing priors since typically, we don't seriously believe that the "true" generative process can be implemented exactly with a small model. Moreover, optimizing (or integrating) over continuous hyperparameters may be easier than optimizing over the discrete space of model sizes. In the end, whichever view we take, Occam's Razor is always at work discouraging overcomplex models.

### Acknowledgements

This work was supported by the Danish Research Councils through the Computational Neural Network Center (CONNECT) and the THOR Center for Neuroinformatics. Thanks to Geoff Hinton for asking a puzzling question which stimulated the writing of this paper.

## Footnotes

[1] We really ought to average together predictions from all models weighted by their probabilities. However if the evidence is strongly peaked, or for practical reasons, we may want to select one as an approximation.

[2] For some models, the limit of an infinite number of parameters is a simple model which can be treated tractably. Two examples are the Gaussian Process limit of Bayesian neural networks [Neal, 1996], and the infinite limit of Gaussian mixture models [Rasmussen, 2000].

[3]Under the prior, the joint density of any (finite) set of outputs $y$ is Gaussian

[4]We choose vague priors by setting $\alpha_1 = \alpha_2 = \beta_1 = \beta_2 = 0.2$ throughout.

[5]Of course, we ought to integrate over $C$, but unfortunately that is difficult.

### References

Jefferys, W. H. & Berger, J. O. (1992) Ockham's Razor and Bayesian Analysis. *Amer. Sci.*, **80**:64–72.

MacKay, D. J. C. (1992) Bayesian Interpolation. *Neural Computation*, **4**(3):415–447.

Neal, R. M. (1996) *Bayesian Learning for Neural Networks*, Lecture Notes in Statistics No. 118, New York: Springer-Verlag.

Rasmussen, C. E. (2000) The Infinite Gaussian Mixture Model, in S. A. Solla, T. K. Leen and K.-R. Müller (editors.), *Adv. Neur. Inf. Proc. Sys. 12*, MIT Press, pp. 554–560.

Smith, A. F. M. & Spiegelhalter, D. J. (1980) Bayes factors and choice criteria for linear models. *J. Roy. Stat. Soc.*, **42**:213–220.
